# Diffeomorphic Dimensionality Reduction

**Christian Walder and Bernhard Schölkopf**
Max Planck Institute for Biological Cybernetics
72076 Tübingen, Germany
`first.last@tuebingen.mpg.de`

## Abstract

This paper introduces a new approach to constructing meaningful lower dimensional representations of sets of data points. We argue that constraining the mapping between the high and low dimensional spaces to be a diffeomorphism is a natural way of ensuring that pairwise distances are approximately preserved. Accordingly we develop an algorithm which diffeomorphically maps the data near to a lower dimensional subspace and then projects onto that subspace. The problem of solving for the mapping is transformed into one of solving for an Eulerian flow field which we compute using ideas from kernel methods. We demonstrate the efficacy of our approach on various real world data sets.

## 1 Introduction

The problem of visualizing high dimensional data often arises in the context of exploratory data analysis. For many real world data sets this is a challenging task, as the spaces in which the data lie are often too high dimensional to be visualized directly. If the data themselves lie on a lower dimensional subspace however, dimensionality reduction techniques may be employed, which aim to meaningfully represent the data as elements of this lower dimensional subspace.

The earliest approaches to dimensionality reduction are the linear methods known as principal components analysis (PCA) and factor analysis (Duda et al., 2000). More recently however, the majority of research has focussed on non-linear methods, in order to overcome the limitations of linear approaches—for an overview and numerical comparison see *e.g.* (Venna, 2007; van der Maaten et al., 2008), respectively. In an effort to better understand the numerous methods which have been proposed, various categorizations have been proposed. In the present case, it is pertinent to make the distinction between methods which focus on properties of the *mapping* to the lower dimensional space, and methods which focus on properties of the *mapped data*, in that space. A canonical example of the latter is multidimensional scaling (MDS), which in its basic form finds the minimizer with respect to $\boldsymbol{y}_1, \boldsymbol{y}_2, \ldots, \boldsymbol{y}_m$ of (Cox & Cox, 1994)

$$\sum_{i,j=1}^{m} \left( \|\boldsymbol{x}_i - \boldsymbol{x}_j\| - \|\boldsymbol{y}_i - \boldsymbol{y}_j\| \right)^2, \tag{1}$$

where here, as throughout the paper, the $\boldsymbol{x}_i \in \mathbb{R}^a$ are input or high dimensional points, and the $\boldsymbol{y}_i \in \mathbb{R}^b$ are output or low dimensional points, so that $b < a$. Note that the above term is a function only of the input points and the corresponding mapped points, and is designed to preserve the pairwise distances of the data set.

The methods which focus on the mapping itself (from the higher to the lower dimensional space, which we refer to as the downward mapping, or the upward mapping which is the converse) are less common, and form a category into which the present work falls. Both auto-encoders (DeMers & Cottrell, 1993) and the Gaussian process latent variable model (GP-LVM) (Lawrence, 2004) also fall into this category, but we focus on the latter as it provides an appropriate transition into the

main part the paper. The GP-LVM places a Gaussian process (GP) prior over each high dimensional component of the upward mapping, and optimizes with respect to the set of low dimensional points—which can be thought of as hyper-parameters of the model—the likelihood of the high dimensional points. Hence the GP-LVM constructs a regular (in the sense of regularization, *i.e.* likely under the GP prior) upward mapping. By doing so, the model guarantees that nearby points in the low dimensional space should be mapped to nearby points in the high dimensional space—an intuitive idea for dimensionality reduction which is also present in the MDS objective (1), above.

The converse is not guaranteed in the original GP-LVM however, and this has lead to the more recent development of the so-called back-constrained GP-LVM (Lawrence & Candela, 2006), which essentially places an additional GP prior over the downward mapping. By guaranteeing in this way that (the modes of the posterior distributions over) both the upward and downward mappings are regular, the back constrained GP-LVM induces something reminiscent of a diffeomorphic mapping between the two spaces. This leads us to the present work, in which we derive our new algorithm, *Diffeomap*, by explicitly casting the dimensionality reduction problem as one of constructing a diffeomorphic mapping between the low dimensional space and the subspace of the high dimensional space on which the data lie.

## 2 Diffeomorphic Mappings and their Practical Construction

In this paper we use the following definition:

**Definition 2.1.** Let $U$ and $V$ be open subsets of $\mathbb{R}^a$ and $\mathbb{R}^b$, respectively. The mapping $F : U \to V$ is said to be a **diffeomorphism** if it is bijective (*i.e.* one to one), smooth (*i.e.* belonging to $C^\infty$), and has a smooth inverse map $F^{-1}$.

We note in passing the connection between this definition, our discussion of the GP-LVM, and dimensionality reduction. The GP-LVM constructs a regular upward mapping (analogous to $F^{-1}$) which ensures that points nearby in $\mathbb{R}^b$ will be mapped to points nearby in $\mathbb{R}^a$, a property referred to as *similarity preservation* in (Lawrence & Candela, 2006). The back constrained GP-LVM simultaneously ensures that the downward mapping (analogous to $F$) is regular, thereby additionally implementing what its authors refer to as *dissimilarity preservation*. Finally, the similarity between smoothness (required of $F$ and $F^{-1}$ in Definition 2.1) and regularity (imposed on the downward and upward mappings by the GP prior in the back constrained GP-LVM) complete the analogy. There is also an alternative, more direct motivation for diffeomorphic mappings in the context of dimensionality reduction, however. In particular, a diffeomorphic mapping has the property that it does not lose any information. That is, given the mapping itself and the lower dimensional representation of the data set, it is always possible to reconstruct the original data.

There has been significant interest from within the image processing community, in the construction of diffeomorphic mappings for the purpose of image warping (Dupuis & Grenander, 1998; Joshi & Miller, 2000; Karaçali & Davatzikos, 2003). The reason for this can be understood as follows. Let $I : U \to \mathbb{R}^3$ represent the RGB values of an image, where $U \subset \mathbb{R}^2$ is the image plane. If we now define the warped version of $I$ to be $I \circ W$, then we can guarantee that the warp is topology preserving, *i.e.* that it does not "tear" the image, by ensuring the $W$ be a diffeomorphism $U \to U$. The following two main approaches to constructing such diffeomorphisms have been taken by the image processing community, the first of which we mention for reference, while the second forms the basis of Diffeomap. It is a notable aside that there seem to be no image warping algorithms analogous to the back constrained GP-LVM, in which regular forward and inverse mappings are simultaneously constructed.

1. Enforcement of the constraint that $|J(W)|$, the determinant of the Jacobian of the mapping, be positive everywhere. This approach has been successfully applied to the problem of warping 3D magnetic resonance images (Karaçali & Davatzikos, 2003), for example, but a key ingredient of that success was the fact that the authors defined the mapping $W$ numerically on a regular grid. For the high dimensional cases relevant to dimensionality reduction however, such a numerical grid is highly computationally unattractive.

2. Recasting the problem of constructing $W$ as an Eulerian flow problem (Dupuis & Grenander, 1998; Joshi & Miller, 2000). This approach is the focus of the next section.

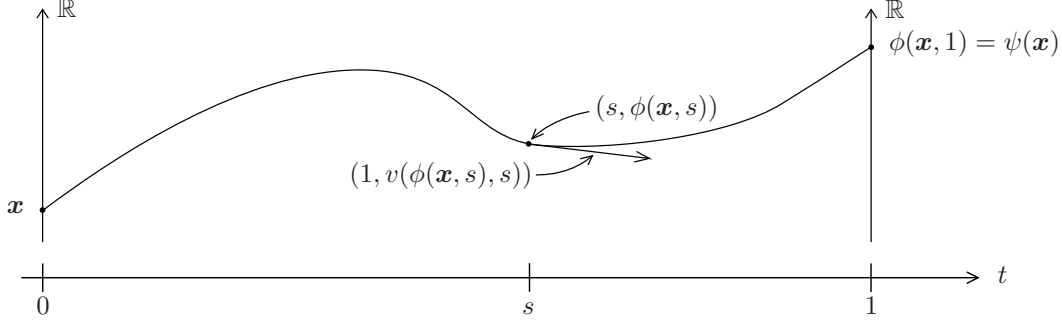

Figure 1: The relationship between $v(\cdot, \cdot)$, $\phi(\cdot, \cdot)$ and $\psi(\cdot)$ for the one dimensional case $\psi : \mathbb{R} \to \mathbb{R}$.

## 2.1 Diffeomorphisms via Flow Fields

The idea here is to indirectly define the mapping of interest, call it $\psi : \mathbb{R}^a \to \mathbb{R}^a$, by way of a "time" indexed velocity field $v : \mathbb{R}^a \times \mathbb{R} \to \mathbb{R}^a$. In particular we write $\psi(\boldsymbol{x}) = \phi(\boldsymbol{x}, 1)$, where

$$\phi(\boldsymbol{x}, t) = \boldsymbol{x} + \int_{s=0}^{t} v(\phi(\boldsymbol{x}, s), s) \mathrm{d}s. \tag{2}$$

This choice of $\phi$ satisfies the following Eulerian transport equation with boundary conditions:

$$\frac{\partial \phi(\boldsymbol{x}, s)}{\partial s} = v(\phi(\boldsymbol{x}, s), s), \quad \phi(\boldsymbol{x}, 0) = \boldsymbol{x}. \tag{3}$$

The role of $v$ is to *transport* a given point $\boldsymbol{x}$ from its original location at time 0 to its mapped location $\phi(\boldsymbol{x}, 1)$ by way of a trajectory whose position and tangent vector at time $s$ are given by $\phi(\boldsymbol{x}, s)$ and $v(\phi(\boldsymbol{x}, s), s)$, respectively (see Figure 1). The point of this construction is that if $v$ satisfies certain regularity properties, then the mapping $\psi$ will be a diffeomorphism. This fact has been proven in a number of places—one particularly accessible example is (Dupuis & Grenander, 1998), where the necessary conditions are provided for the three dimensional case along with a proof that the induced mapping is a diffeomorphism. Generalizing the result to higher dimensions is straightforward—this fact is stated in (Dupuis & Grenander, 1998) along with the basic idea of how to do so.

We now offer an intuitive argument for the result. Consider Figure 1, and imagine adding a new starting point $\boldsymbol{x}'$, along with its associated trajectory. It is clear that for the mapping $\psi$ to be a diffeomorphism, then for any such pair of points $\boldsymbol{x}$ and $\boldsymbol{x}'$, the associated trajectories must not collide. This is because the two trajectories would be identical after the collision, $\boldsymbol{x}$ and $\boldsymbol{x}'$ would map to the same point, and hence the mapping would not be invertible. But if $v$ is sufficiently regular then such collisions cannot occur.

## 3 Diffeomorphic Dimensionality Reduction

The framework of Eulerian flow fields which we have just introduced provides an elegant means of constructing diffeomorphic mappings $\mathbb{R}^a \to \mathbb{R}^a$, but for dimensionality reduction we require additional ingredients, which we now introduce. The basic idea is to construct a diffeomorphic mapping in such a way that it maps our data set near to a subspace of $\mathbb{R}^a$, and then to project onto this subspace. The subspace we use, call it $S_b$, is the $b$-dimensional one spanned by the first $b$ canonical basis vectors of $\mathbb{R}^a$. Let $P_{(a \to b)} : \mathbb{R}^a \to \mathbb{R}^b$ be the projection operator which extracts the first $b$ components of the vector it is applied to, *i.e.*

$$P_{(a \to b)} \boldsymbol{x} = (I \ Z) \, \boldsymbol{x}, \tag{4}$$

where $I \in \mathbb{R}^{a \times a}$ is the identity matrix and $Z \in \mathbb{R}^{a \times b - a}$ is a matrix of zeros. We can now write the mapping $\varphi : \mathbb{R}^a \to \mathbb{R}^b$ which we propose for dimensionality reduction as

$$\varphi(\boldsymbol{x}) = P_{(a \to b)} \phi(\boldsymbol{x}, 1), \tag{5}$$

where $\phi$ is given by (2). We choose each component of $v$ at each time to belong to a reproducing kernel Hilbert Space (RKHS) $\mathcal{H}$, so that $v(\cdot, t) \in \mathcal{H}^a, t \in [0, 1]$. If we define the norm[1]

$$\|v(\cdot, t)\|_{\mathcal{H}^a}^2 \triangleq \sum_{j=1}^{a} \left\| [v(\cdot, t)]_j \right\|_{\mathcal{H}}^2, \tag{6}$$

then $\|v(\cdot, t)\|_{\mathcal{H}^a}^2 < \infty, \forall t \in [0, 1]$ is a sufficient condition which guarantees that $\psi$ is a diffeo-morphism, provided that some technical conditions are satisfied (Dupuis & Grenander, 1998; Joshi & Miller, 2000). In particular $v$ need not be regular in its second argument. For dimensionality reduction we propose to construct $v$ as the minimizer of

$$O = \lambda \int_{t=0}^{1} \|v(\cdot, t)\|_{\mathcal{H}^d}^2 \, \mathrm{d}t + \sum_{j=1}^{m} L\left(\psi(\boldsymbol{x}_j)\right), \tag{7}$$

where $\lambda \in \mathbb{R}^+$ is a regularization parameter. Here, $L$ measures the squared distance to our $b$ dimensional linear subspace of interest $S_b$, *i.e.*

$$L(\boldsymbol{x}) = \sum_{d=b+1}^{a} [\boldsymbol{x}]_d^2. \tag{8}$$

Note that this places special importance on the first $b$ dimensions of the input space of interest— accordingly we make the natural and important preprocessing step of applying PCA such that as much as possible of the variance of the data is captured in these first $b$ dimensions.

## 3.1 Implementation

One can show that the minimizer in $v$ of (7) takes the form

$$[v(\cdot, t)]_d = \sum_{j=1}^{m} [\alpha_d(t)]_j \, k(\phi(\boldsymbol{x}_j, t), \cdot), \quad d = 1 \dots a, \tag{9}$$

where $k$ is the reproducing kernel of $\mathcal{H}$ and $\alpha_d$ is a function $[0, 1] \to \mathbb{R}^m$. This was proven directly for a similar specific case (Joshi & Miller, 2000), but we note in passing that it follows immediately from the celebrated representer theorem of RKHS's (Schölkopf et al., 2001), by considering a fixed time $t$. Hence, we have simplified the problem of determining $v$ to one of determining $m$ trajectories $\phi(\boldsymbol{x}_j, \cdot)$. This is because not only does (9) hold, but we can use standard manipulations (in the context of kernel ridge regression, for example) to determine that for a given set of such trajectories,

$$\alpha_d(t) = K(t)^{-1} u_d(t), \quad d = 1, 2, \dots, a, \tag{10}$$

where $t \in [0, 1]$, $K(t) \in \mathbb{R}^{m \times m}$, $u_d(t) \in \mathbb{R}^m$ and we have let $[K(t)]_{j,k} = k(\phi(\boldsymbol{x}_j, t), \phi(\boldsymbol{x}_k, t))$ along with $[u_d(t)]_j = \partial_t \phi(\boldsymbol{x}_j, t)$. Note that the invertibility of $K(t)$ is guaranteed for certain kernel functions (including the Gaussian kernel which we employ in all our Experiments, see Section 4), provided that the set $\phi(\boldsymbol{x}_j, t)$ are distinct. Hence, one can verify using (9), (10) and the reproducing property of $k$ in $\mathcal{H}$ (*i.e.* the fact that $\langle f, k(\boldsymbol{x}, \cdot) \rangle_{\mathcal{H}} = f(\boldsymbol{x}), \forall f \in \mathcal{H}$), that for the optimal $v$,

$$\|v(\cdot, t)\|_{\mathcal{H}^a}^2 = \sum_{d=1}^{a} u_d(t)^\top K(t)^{-1} u_d(t). \tag{11}$$

This allows us to write our objective (7) in terms of the $m$ trajectories mentioned above:

$$O = \lambda \int_{t=0}^{1} \sum_{d=1}^{a} u_d(t)^\top K(t)^{-1} u_d(t) + \sum_{j=1}^{m} \sum_{d=b+1}^{a} [\phi(\boldsymbol{x}_j, 1)]_d^2. \tag{12}$$

So far no approximations have been made, and we have constructed an optimal finite dimensional basis for $v(\cdot, t)$. The second argument of $v$ is not so easily dealt with however, so as an approximate by discretizing the interval $[0, 1]$. In particular, we let $t_k = k\delta, k = 0, 1, \dots, p$, where $\delta = 1/p$, and make the approximation $\partial_{t=t_k} \phi(\boldsymbol{x}_j, t) = (\phi(\boldsymbol{x}_j, t_k) - \phi(\boldsymbol{x}_j, t_{k-1})) / \delta$. By making the further

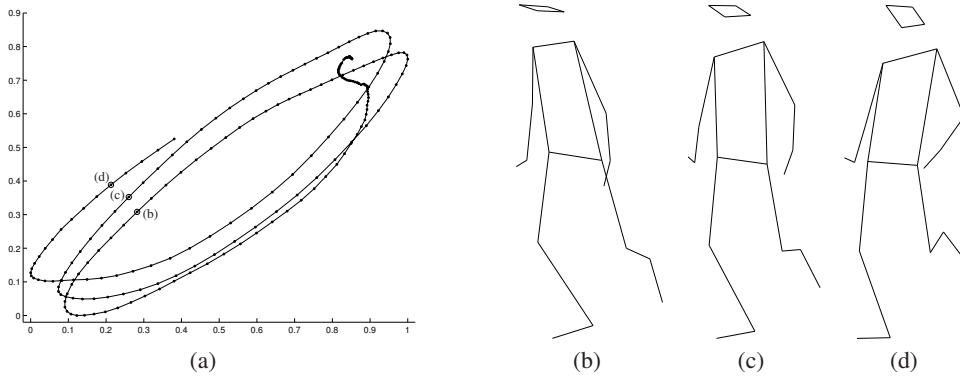

(a)              (b)        (c)        (d)

Figure 2: Dimensionality reduction of motion capture data. (a) The data mapped from 102 to 2 dimensions using Diffeomap (the line shows the temporal order in which the input data were recorded). (b)-(d) Three rendered input points corresponding to the marked locations in (a).

approximation $\int_{t=t_{k-1}}^{t_k} K(t)^{-1}\mathrm{d}t = \delta K(t_{k-1})^{-1}$, and substituting into (12) we obtain the first form of our problem which is finite dimensional and hence readily optimized, *i.e.* the minimization of

$$\frac{\lambda}{\delta} \sum_{d=1}^{a}\sum_{k=1}^{p} (\Phi_{k,d} - \Phi_{k-1,d})^{\top} K(t_k)^{-1} (\Phi_{k,d} - \Phi_{k-1,d}) + \sum_{d=a+1}^{b} \|\Phi_{p,d}\|^2 \tag{13}$$

with respect to $\Phi_{k,d} \in \mathbb{R}^m$ for $k = 1, 2, \ldots, p$ and $d = 1, 2, \ldots, a$, where $[\Phi_{k,d}]_j = [\phi(\boldsymbol{x}_j, t_k)]_d$.

### 3.2 A Practical Reduced Set Implementation

A practical problem with (13) is the computationally expensive matrix inverse. In practice we reduce this burden by employing a reduced set expansion which replaces the sum over $1, 2, \ldots, m$ in (9) with a sum over a randomly selected subset $\mathcal{I}$, thereby using $|\mathcal{I}| = n$ basis functions to represent $v(\cdot, t)$. In this case it is possible to show using the reproducing property of $k(\cdot, \cdot)$ that the resulting objective function is identical to (13), but with the matrix $K(t_k)^{-1}$ replaced by the expression

$$K_{m,n} (K_{n,m}K_{m,n})^{-1} K_{n,n} (K_{n,m}K_{m,n})^{-1} K_{n,m}, \tag{14}$$

where $K_{m,n} = K_{n,m}^{\top} \in \mathbb{R}^{m \times n}$ is the sub-matrix of $K(t_k)$ formed by taking all of the rows, but only those columns given by $\mathcal{I}$. Similarly, $K_{n,n} \in \mathbb{R}^{n \times n}$ is the square sub-matrix of $K(t_k)$ formed by taking a subset of both the rows and columns, namely those given by $\mathcal{I}$. For optimization we also use the gradients of the above expression, the derivation of which we have omitted for brevity. Note however that by factorizing appropriately, the computation of the objective function and its gradients can be performed with an asymptotic time complexity of $n^2(m + a)$.

## 4 Experiments

It is difficult to objectively compare dimensionality reduction algorithms, as there is no universally agreed upon measure of performance. Algorithms which are generalizations or variations of older ones may be compared side by side with their predecessors, but this is not the case with our new algorithm, Diffeomap. Hence, in this section we attempt to convince the reader of the utility of our approach by visually presenting our results on as many and as varied realistic problems as space permits, while providing pointers to comparable results from other authors. For all experiments we fixed the parameters which trade off between computational speed and accuracy, *i.e.* we set the temporal resolution $p = 20$, and the number of basis functions $n = 300$. We used a Gaussian kernel function $k(\boldsymbol{x}, \boldsymbol{y}) = \exp\left(-\|\boldsymbol{x} - \boldsymbol{y}\|^2/(2\sigma^2)\right)$, and tuned the $\sigma$ parameter manually along with the regularization parameter $\lambda$. For optimization we used a conjugate gradient type method[2] fixed to 1000 iterations and with starting point $[\Phi_{k,d}]_j = [\boldsymbol{x}_j]_d$, $k = 1, 2, \ldots p$.

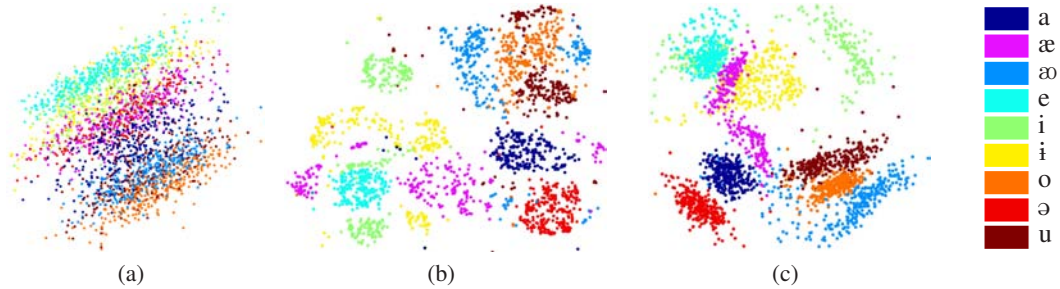

| | |
|---|---|
| (a) | a |
| | æ |
| | ɐ |
| | e |
| | i |
| | ɨ |
| | o |
| | ə |
| | u |

(a)                    (b)                    (c)

Figure 3: Vowel data mapped from 24 to 2 dimensions using (a) PCA and (b)-(c) Diffeomap. Plots (b) and (c) differ only in the parameter settings of Diffeomap, with (b) corresponding to minimal one nearest neighbor errors in the low dimensional space—see Section 4.2 for details.

## 4.1 Motion Capture Data

The first data set we consider consists of the coordinates in $\mathbb{R}^3$ of a set of markers placed on a person breaking into a run, sampled at a constant frequency, resulting in $m = 217$ data points in $a = 102$ dimensions, which we mapped to $b = 2$ dimensions using Diffeomap (see Figure 2). This data set is freely available from `http://accad.osu.edu/research/mocap/mocap_data.htm` as *Figure 1 Run*, and was also considered in (Lawrence & Candela, 2006), where it was shown that while the original GP-LVM fails to correctly discover the periodic component of the sequence, the back constrained version maps poses in the same part of the subject's step cycle nearby to each other, while simultaneously capturing variations in the inclination of the subject. Diffeomap also succeeded in this sense, and produced results which are competitive with those of the back constrained GP-LVM.

## 4.2 Vowel Data

In this next example we consider a data set of $a = 24$ features (cepstral coefficients and delta cepstral coefficients) of a single speaker performing nine different vowels 300 times per vowel, acquired as training data for a vocal joystick system (Bilmes & et.al., 2006), and publicly available in pre-processed form from `http://www.dcs.shef.ac.uk/~neil/fgplvm/`. Once again we used Diffeomap to map the data to $b = 2$ dimensions, as depicted in Figure 3. We also depict the poor result of linear PCA, in order to rule out the hypothesis that it is merely the PCA based initialization of Diffeomap (mentioned after equation (8) on page 4) which does most of the work.

The results in Figure 3 are directly comparable to those provided in (Lawrence & Candela, 2006) for the GP-LVM, back constrained GP-LVM, and Isomap (Tenenbaum et al., 2000). Visually, the Diffeomap result appears to be superior to those of the GP-LVM and Isomap, and comparable to the back constrained GP-LVM. We also measured the performance of a one nearest neighbor classifier applied to the mapped data in $\mathbb{R}^2$. For the best choice of the parameters $\sigma$ and $\lambda$, Diffeomap made 140 errors, which is favorable to the figures quoted for Isomap (458), the GP-LVM (226) and the back constrained GP-LVM (155) in (Lawrence & Candela, 2006). We emphasize however that this measure of performance is at best a rough one, since by manually varying our choice of the parameters $\sigma$ and $\lambda$, we were able to obtain a result (Figure 3 (c)) which, although leads to a significantly higher number of such errors (418), is arguably superior from a qualitative perspective to the result with minimal errors (Figure 3 (b)).

## 4.3 USPS Handwritten Digits

We now consider the USPS database of handwritten digits (Hull, 1994). Following the methodology of the stochastic neighbor embedding (SNE) and GP-LVM papers (Hinton & Roweis, 2003; Lawrence, 2004), we take 600 images per class from the five classes corresponding to digits 0, 1, 2, 3, 4. Since the images are in gray scale and a resolution of 16 by 16 pixels, this results in a data set of $m = 3000$ examples in $a = 256$ dimensions, which we again mapped to $b = 2$ dimensions as depicted in Figure 4. The figure shows the individual points color coded according to class, along

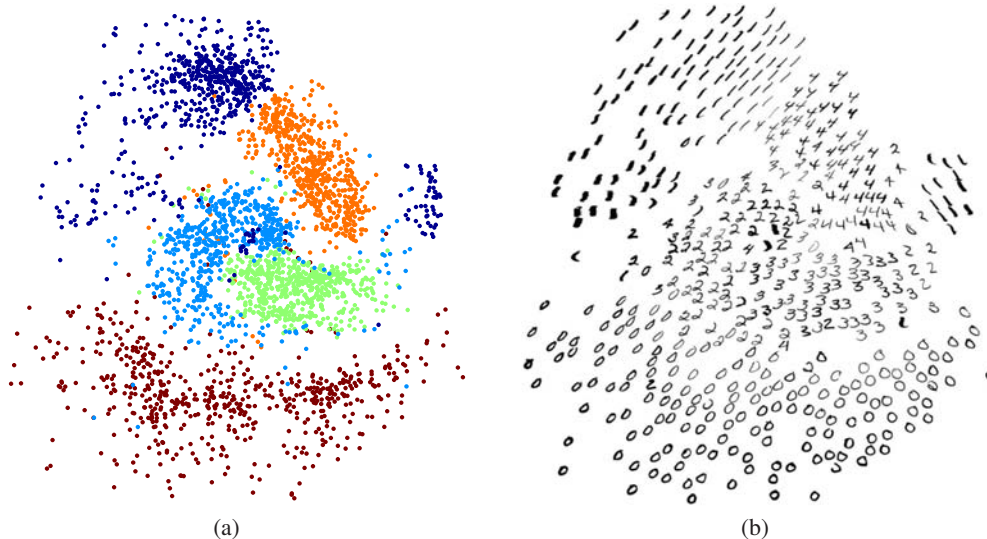

<center>(a)</center> <center>(b)</center>

Figure 4: USPS handwritten digits 0-4 mapped to 2 dimensions using Diffeomap. (a) Mapped points color coded by class label. (b) A composite image of the mapped data—see Section 4.3 for details.

with a composite image formed by sequentially drawing each digit in random order at its mapped location, but only if it would not obscure a previously drawn digit. Diffeomap manages to arrange the data in a manner which reveals such image properties as digit angle and stroke thickness. At the same time the classes are reasonably well separated, with the exception of the ones which are split into two clusters depending on the angle. Although unfortunate, we believe that this splitting can be explained by the fact that (a) the left- and right-pointing ones are rather dissimilar in input space, and (b) the number of fairly vertical ones which could help to connect the left- and right-pointing ones is rather small. Diffeomap seems to produce a result which is superior to that of the GP-LVM (Lawrence, 2004), for example, but may be inferior to that of the SNE (Hinton & Roweis, 2003). We believe this is due to the fact that the nearest neighbor graph used by SNE is highly appropriate to the USPS data set. This is indicated by the fact that a nearest neighbor classifier in the 256 dimensional input space is known to perform strongly, with numerous authors having reported error rates of less than 5% on the ten class classification problem.

## 4.4 NIPS Text Data

Finally, we present results on the text data of papers from the NIPS conference proceedings volumes 0-12, which can be obtained from `http://www.cs.toronto.edu/~roweis/data.html`. This experiment is intended to address the natural concern that by working in the input space rather than on a nearest neighbor graph, for example, Diffeomap may have difficulty with very high dimensional data. Following (Hinton & Roweis, 2003; Song et al., 2008) we represent the data as a word frequency *vs.* document matrix in which the author names are treated as words but weighted up by a factor 20 (*i.e.* an author name is worth 20 words). The result is a data set of $m = 1740$ papers represented in $a = 13649$ words $+ 2037$ authors $= 15686$ dimensions. Note however that the input dimensionality is effectively reduced by the PCA preprocessing step to $m - 1 = 1739$, that being the rank of the centered covariance matrix of the data.

As this data set is difficult to visualize without taking up large amounts of space, we have included the results in the supplementary material which accompanies our NIPS submission. In particular, we provide a first figure which shows the data mapped to $b = 2$ dimensions, with certain authors (or groups of authors) color coded—the choice of authors and their corresponding color codes follows precisely those of (Song et al., 2008). A second figure shows a plain marker drawn at the mapped locations corresponding to each of the papers. This second figure also contains the paper title and authors of the corrsponding papers however, which are revealed when the user moves the mouse over the marked locations. Hence, this second figure allows one to browse the NIPS collection contextually. Since the mapping may be hard to judge, we note in passing that the correct classification rate of a one nearest neighbor classifier applied to the result of Diffeomap was 48%, which compares favorably to the rate of 33% achieved by linear PCA (which we use for preprocessing). To compute this score we treated authors as classes, and considered only those authors who were color coded both in our supplementary figure and in (Song et al., 2008).

## 5   Conclusion

We have presented an approach to dimensionality reduction which is based on the idea that the mapping between the lower and higher dimensional spaces should be diffeomorphic. We provided a justification for this approach, by showing that the common intuition that dimensionality reduction algorithms should approximately preserve pairwise distances of a given data set is closely related to the idea that the mapping induced by the algorithm should be a diffeomorphism. This realization allowed us to take advantage of established mathematical machinery in order to convert the dimensionality reduction problem into a so called Eulerian flow problem, the solution of which is guaranteed to generate a diffeomorphism. Requiring that the mapping and its inverse both be smooth is reminiscent of the GP-LVM algorithm (Lawrence & Candela, 2006), but has the advantage in terms of statistical strength that we need not separately estimate a mapping in each direction. We showed results of our algorithm, Diffeomap, on a relatively small motion capture data set, a larger vowel data set, the USPS image data set, and finally the rather high dimensional data set derived from the text corpus of NIPS papers, with successes in all cases. Since our new approach performs well in practice while being significantly different to all previous approaches to dimensionality reduction, it has the potential to lead to a significant new direction in the field.

## Footnotes

[1]Square brackets w/ subscripts denote matrix elements, and colons denote entire rows or columns.

[2]Carl Rasmussen's `minimize.m`, which is freely available from `http://www.kyb.mpg.de/~carl`.

## References

Bilmes, J., & et.al. (2006). The Vocal Joystick. *Proc. IEEE Intl. Conf. on Acoustic, Speech and Signal Processing*. Toulouse, France.

Cox, T., & Cox, M. (1994). *Multidimensional scaling*. London, UK: Chapman & Hall.

DeMers, D., & Cottrell, G. (1993). Non-linear dimensionality reduction. *NIPS 5* (pp. 580–587). Morgan Kaufmann, San Mateo, CA.

Duda, R. O., Hart, P. E., & Stork, D. G. (2000). *Pattern classification*. New York: Wiley. 2nd Edition.

Dupuis, P., & Grenander, U. (1998). Variational problems on flows of diffeomorphisms for image matching. *Quarterly of Applied Mathematics*, *LVI*, 587–600.

Hinton, G., & Roweis, S. (2003). Stochastic neighbor embedding. In S. T. S. Becker and K. Obermayer (Eds.), *Advances in neural information processing systems 15*, 833–840. Cambridge, MA: MIT Press.

Hull, J. J. (1994). A database for handwritten text recognition research. *IEEE Trans. Pattern Anal. Mach. Intell.*, *16*, 550–554.

Joshi, S. C., & Miller, M. I. (2000). Landmark matching via large deformation diffeomorphisms. *IEEE Transactions on Image Processing*, *9*, 1357–1370.

Karaçali, B., & Davatzikos, C. (2003). Topology preservation and regularity in estimated deformation fields. *Information Processing in Medical Imaging* (pp. 426–437).

Lawrence, N. D. (2004). Gaussian process latent variable models for visualisation of high dimensional data. In S. Thrun, L. Saul and B. Schölkopf (Eds.), *Nips 16*. Cambridge, MA: MIT Press.

Lawrence, N. D., & Candela, J. Q. (2006). Local distance preservation in the GP-LVM through back constraints. In *International conference on machine learning*, 513–520. ACM.

Schölkopf, B., Herbrich, R., & Smola, A. J. (2001). A generalized representer theorem. *Proc. of the 14th Annual Conf. on Computational Learning Theory* (pp. 416–426). London, UK: Springer-Verlag.

Song, L., Smola, A., Borgwardt, K., & Gretton, A. (2008). Colored maximum variance unfolding. In J. Platt, D. Koller, Y. Singer and S. Roweis (Eds.), *Nips 20*, 1385–1392. Cambridge, MA: MIT Press.

Tenenbaum, J. B., de Silva, V., & Langford, J. C. (2000). A global geometric framework for nonlinear dimensionality reduction. *Science*, *290*, 2319–2323.

van der Maaten, L. J. P., Postma, E., & van den Herik, H. (2008). Dimensionality reduction: A comparative review. In T. Ertl (Ed.), *Submitted to neurocognition*. Elsevier.

Venna, J. (2007). *Dimensionality reduction for visual exploration of similarity structures*. Doctoral dissertation, Helsinki University of Technology.

